# Clustering by Nonnegative Matrix Factorization Using Graph Random Walk

**Zhirong Yang, Tele Hao, Onur Dikmen, Xi Chen and Erkki Oja**
Department of Information and Computer Science
Aalto University, 00076, Finland
{zhirong.yang,tele.hao,onur.dikmen,xi.chen,erkki.oja}@aalto.fi

## Abstract

Nonnegative Matrix Factorization (NMF) is a promising relaxation technique for clustering analysis. However, conventional NMF methods that directly approximate the pairwise similarities using the least square error often yield mediocre performance for data in curved manifolds because they can capture only the immediate similarities between data samples. Here we propose a new NMF clustering method which replaces the approximated matrix with its smoothed version using random walk. Our method can thus accommodate farther relationships between data samples. Furthermore, we introduce a novel regularization in the proposed objective function in order to improve over spectral clustering. The new learning objective is optimized by a multiplicative Majorization-Minimization algorithm with a scalable implementation for learning the factorizing matrix. Extensive experimental results on real-world datasets show that our method has strong performance in terms of cluster purity.

## 1 Introduction

Clustering analysis as a discrete optimization problem is usually NP-hard. Nonnegative Matrix Factorization (NMF) as a relaxation technique for clustering has shown remarkable progress in the past decade (see e.g. [9, 4, 2, 26]). In general, NMF finds a low-rank approximating matrix to the input nonnegative data matrix, where the most popular approximation criterion or divergence in NMF is the Least Square Error (LSE). It has been shown that certain NMF variants with this divergence measure are equivalent to k-means, kernel k-means, or spectral graph cuts [7]. In addition, NMF with LSE can be implemented efficiently by existing optimization methods (see e.g. [16]).

Although popularly used, previous NMF methods based on LSE often yield mediocre performance for clustering, especially for data that lie in a curved manifold. In clustering analysis, the cluster assignment is often inferred from pairwise similarities between data samples. Commonly the similarities are calculated based on Euclidean distances. For data in a curved manifold, only local Euclidean distances are reliable and similarities between non-neighboring samples are usually set to zero, which yields a sparse input matrix to NMF. If the LSE is directly used in approximation to such a similarity matrix, a lot of learning effort will be wasted due to the large majority of zero entries. The same problem occurs for clustering nodes of a sparse network.

In this paper we propose a new NMF method for clustering such manifold data or sparse network data. Previous NMF clustering methods based on LSE used an approximated matrix that takes only similarities within immediate neighborhood into account. Here we consider multi-step similarities between data samples using graph random walk, which has shown to be an effective smoothing approach for finding global data structures such as clusters. In NMF the smoothing can reduce the sparsity gap in the approximation and thus ease cluster analysis. We name the new method *NMF using graph Random walk* (NMFR).

In implementation, we face two obstacles when the input matrix is replaced by its random walk version: (1) the performance of unconstrained NMFR remains similar to classical spectral clustering because smoothing that manipulates eigenvalues of Laplacian of the similarity graph does not change the eigensubspace; (2) The similarities by random walk require inverting an $n \times n$ matrix for $n$ data samples. Explicit matrix inversion is infeasible for large datasets. To overcome the above obstacles, we employ (1) a regularization technique that supplements the orthogonality constraint for better clustering, and (2) a more scalable fixed-point algorithm to calculate the product of the inverted matrix and the factorizing matrix.

We have conducted extensive experiments for evaluating the new method. The proposed algorithm is compared with nine other state-of-the-art clustering approaches on a large variety of real-world datasets. Experimental results show that with only simple initialization NMFR performs pretty robust across 46 clustering tasks. The new method achieves the best clustering purity for 36 of the selected datasets, and nearly the best for the rest. In particular, NMFR is remarkably superior to the other methods for large-scale manifold data from various domains.

In the remaining, we briefly review some related work of clustering by NMF in Section 2. In Section 3 we point out a major drawback in previous NMF methods with least square error and present our solution. Experimental settings and results are given in Section 4. Section 5 concludes the paper and discusses potential future work.

## 2  Pairwise Clustering by NMF

Cluster analysis or clustering is the task of assigning a set of data samples into groups (called clusters) so that the objects in the same cluster are more similar to each other than to those in other clusters. Denote $\mathbb{R}_+ = \mathbb{R} \cup \{0\}$. The pairwise similarities between $n$ data samples can be encoded in an undirected graph with adjacency matrix $S \in \mathbb{R}_+^{n \times n}$. Because clustered data tend to have higher similarities within clusters and lower similarities between clusters, the similarity matrix in visualization has nearly diagonal blockwise looking if we sort the rows and columns by clusters. Such structure motivated approximative low-rank factorization of $S$ by the cluster indicator matrix $U \in \{0, 1\}^{n \times r}$ for $r$ clusters: $S \approx UU^T$, where $U_{ik} = 1$ if the $i$-th sample is assigned to the $k$-th cluster and 0 otherwise. Moreover, clusters of balanced sizes are desired in most clustering tasks. This can be achieved by suitable normalization of the approximating matrix. A common way is to normalize $U_{ik}$ by $M_{ik} = U_{ik}/\sqrt{\sum_j U_{jk}}$ such that $M^T M = I$ and $\sum_i (MM^T)_{ij} = 1$ (see e.g. [6, 7, 27]).

However, directly optimizing over $U$ or $M$ is difficult due to discrete solution space, which usually leads to an NP-hard problem. Continuous relaxation is thus needed to ease the optimization. One of the popular choices is nonnegativity and orthogonality constraint combination [11, 23]. That is, we replace $M$ with $W$ where $W_{ik} \geq 0$ and $W^T W = I$. In this way, each row of $W$ has only one non-zero entry because the non-zero parts of two nonnegative and orthogonal vectors do not overlap. Some other Nonnegative Matrix Factorization (NMF) relaxations exist, for example, the kernel Convex NMF [9] and its special case Projective NMF [23], as well as the relaxation by using a left-stochastic matrix [2].

A commonly used divergence that measures the approximation error is the squared Euclidean distance or Frobenius norm [15, 13]. The NMF objective to be minimized thus becomes

$$\|S - WW^T\|_F^2 = \sum_{ij} \left[ S_{ij} - \left( WW^T \right)_{ij} \right]^2 . \tag{1}$$

The above least square error objective is widely used because we have better understanding of its algebra and geometric properties. For example, Zhao et al. [13] showed that the multiplicative optimization algorithm for the above Symmetric NMF (SNMF) problem is guaranteed to converge to a local minimum if $S$ is positive semi-definite. Furthermore, SNMF with orthogonality has tight connection to classical objectives such as kernel k-means and normalized cuts [7, 23]. In this paper, we choose this divergence also because it is the sole one in $\alpha\beta$-divergence family [5] that involves only the product $SW$ instead of $S$ itself in the gradient. As we shall see in Section 3.2, this property enables a scalable implementation of gradient-based optimization algorithm.

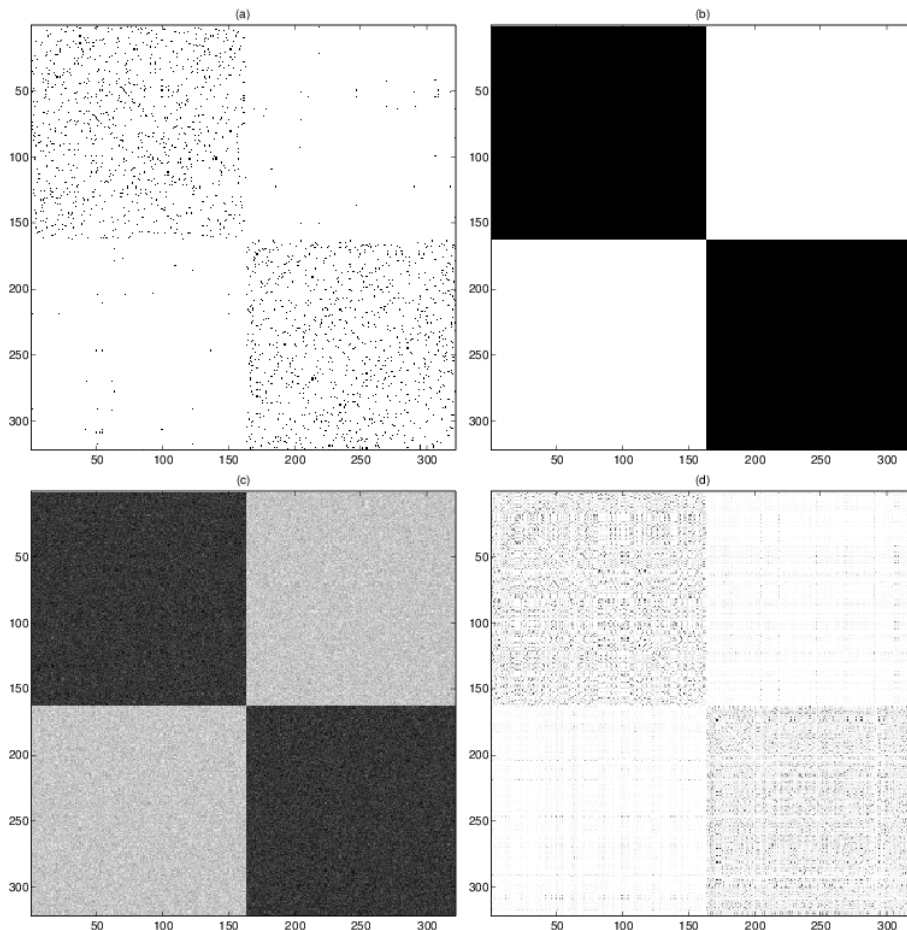

Figure 1: Illustration of clustering the *SEMEION* handwritten digit dataset by NMF based on LSE: (a) the symmetrized 5-NN graph, (b) the correct clusters to be found, (c) the ideally assumed data that suits the least square error, (d) the smoothed input by using graph random walk. The matrix entries are visualized as image pixels. Darker pixels represent higher similarities. For clarity we show only the subset of digits "2" and "3". In this paper we show that because (d) is "closer" to (c) than (a), it is easier to find correct clusters using (d)≈(b) instead of (a)≈(b) by NMF with LSE .

## 3   NMF Using Graph Random Walk

There is a serious drawback in previous NMF clustering methods using least square errors. When minimizing $\|S - \widehat{S}\|_F^2$ for given $S$, the approximating matrix $\widehat{S}$ should be diagonal blockwise for clustering analysis, as shown in Figure 1 (b). Correspondingly, the ideal input $S$ for LSE should look like Figure 1 (c) because the underlying distribution of LSE is Gaussian.

However, the similarity matrix of real-world data often occurs differently from the ideal case. In many clustering tasks, the raw features of data are usually weak. That is, the given distance measure between data points, such as the Euclidean distance, is only valid in a small neighborhood. The similarities calculated from such distances are thus sparse, where the similarities between non-neighboring samples are usually set to zero. For example, symmetrized $K$-nearest-neighbor (K-NN) graph is a popularly used similarity input. Therefore, similarity matrices in real-world clustering tasks often look like Figure 1 (a), where the non-zero entries are much sparser than the ideal case.

It is a mismatch to approximate a sparse similarity matrix by a dense diagonal blockwise matrix using LSE. Because squared Euclidean distance is a symmetric metric, the learning objective can be dominated by the approximation to the majority of zero entries, which is undesired for finding correct cluster assignments. Although various matrix factorization schemes and factorizing matrix

constraints have been proposed for NMF, little research effort has been made to overcome the above mismatch.

In this work we present a different way to formalize NMF for clustering to reduce the sparsity gap between input and output matrices. Instead of approximation to the sparse input $S$, which only encodes the immediate similarities between data samples, we propose to approximate a smoothed version of $S$ which takes farther relationships between data samples into account. Graph random walk is a common way to implement multi-step similarities. Denote $Q = D^{-1/2}SD^{-1/2}$ the normalized similarity matrix, where $D$ is a diagonal matrix with $D_{ii} = \sum_j S_{ij}$. The similarities between data nodes using $j$ steps are given by $(\alpha Q)^j$, where $\alpha \in (0,1)$ is a decay parameter controlling the random walk extent. Summing over all possible numbers of steps gives $\sum_{j=0}^{\infty} (\alpha Q)^j = (I - \alpha Q)^{-1}$. We thus propose to replace $S$ in Eq. (1) with

$$A = c^{-1}(I - \alpha Q)^{-1}, \tag{2}$$

where $c = \sum_{ij} \left[ (I - \alpha Q)^{-1} \right]_{ij}$ is a normalizing factor. Here the parameter $\alpha$ controls the smoothness: a larger $\alpha$ tends to produce smoother $A$ while a smaller one makes $A$ concentrate on its diagonal. A smoothed approximated matrix $A$ is shown in Figure 1 (d), from which we can see the sparsity gap to the approximating matrix is reduced.

Just smoothing the input matrix by random walk is not enough, as we are presented with two difficulties. First, random walk only alters the spectrum of $Q$, while the eigensubspaces of $A$ and $Q$ are the same. Smoothing therefore does not change the result of clustering algorithms that operate on the eigenvectors (e.g. [20]). If we simply replace $S$ by $A$ in Eq. (1), the resulting $W$ is often the same as the leading eigenvectors of $Q$ up to an $r \times r$ rotation. That is, smoothing by random walk itself can bring little improvement unless we impose extra constraints or regularization. Second, explicitly calculating $A$ is infeasible because when $S$ is large and sparse, $A$ is also large but dense. This requires a more careful design of a scalable optimization algorithm. Below we present solutions to overcome these two difficulties in Sections 3.1 and 3.2, respectively.

## 3.1 Learning Objective

Minimizing $\|A - WW^T\|_F^2$ over $W$ subject to $W^T W = I$ is equivalent to maximizing $\text{Tr}\left(W^T AW\right)$. To improve over spectral clustering, we propose to regularize the trace maximization by an extra penalty term on $W$. The new optimization problem for pairwise clustering is:

$$\underset{W \geq 0}{\text{minimize}} \quad \mathcal{J}(W) = -\text{Tr}\left(W^T AW\right) + \lambda \sum_i \left( \sum_k W_{ik}^2 \right)^2 \tag{3}$$

$$\text{subject to} \ \ W^T W = I, \tag{4}$$

where $\lambda > 0$ is the tradeoff parameter. We find that $\lambda = \frac{1}{2r}$ works well in this work.

The extra penalty term collaborates with the orthogonality constraint for pairwise clustering, which is justified by two interpretations.

- It emphasizes off-diagonal correlation in the trace. Because $\sum_i \left( \sum_k W_{ik}^2 \right)^2 = \sum_i \left( WW^T \right)_{ii}^2$, the minimization tends to reduce the diagonal magnitude in the approximating matrix. This is desired because self-similarities usually give little information for grouping data samples. Given the constraints $W \geq 0$ and $W^T W = I$, it is beneficial to push the magnitudes in $WW^T$ off-diagonal for maximizing the correlation to similarities between different data samples.

- It tends to equalize the norms of $W$ rows. To see this, let us write $a_i \equiv \sum_k W_{ik}^2$ for brevity. Because $\sum_i a_i = r$ is constant, minimizing $\sum_i a_i^2$ actually maximizing $\sum_{ij:i \neq j} a_i a_j$. The maximum is achieved when $\{a_i\}_{i=1}^n$ are equal. Originally, the nonnegativity and orthogonality constraint combination only guarantees that each row of $W$ has one non-zero entry, though norms of different $W$ rows can be diverse. The equalization by the proposed penalty term thus well supplements the nonnegativity and orthogonality constraints and, as a whole, provides closer relaxation to the normalized cluster indicator matrix $M$.

---

**Algorithm 1** Large-Scale Relaxed Majorization and Minimization Algorithm for $W$

---

**Input:** similarity matrix $S$, random walk extent $\alpha \in (0,1)$, number of clusters $r$, nonnegative initial guess of $W$.
**repeat**
    Calculate $c$=IterativeTracer($Q,\alpha, e$).
    Calculate $G$=IterativeSolver($Q,\alpha, W$).
    Update $W$ by Eq. (5), using $c^{-1}G$ in place of $AW$.
**until** $W$ converges
Discretized $W$ to cluster indicator matrix $U$
**Output:** $U$.

**function** ITERATIVETRACER($Q, \alpha, W$)
    $F$=IterativeSolver($Q,\alpha, W$)
    **return** $\text{Tr}(W^T F)$
**end function**

**function** ITERATIVESOLVER($Q, \alpha, W$)
    Initialize $F = W$
    **repeat**
        Update $F \leftarrow \alpha Q F + (1 - \alpha)W$
    **until** $F$ converges
    **return** $F/(1 - \alpha)$
**end function**

---

## 3.2 Optimization

The optimization algorithm is developed by following the procedure in [24, 26]. Introducing Lagrangian multipliers $\{\Lambda_{kl}\}$ for the orthogonality constraint, we have the augmented objective $\mathcal{L}(W, \Lambda) = \mathcal{J}(W) + \text{Tr}\left[\Lambda\left(W^T W - I\right)\right]$. Using the Majorization-Minimization development procedure in [24, 26], we can obtain the preliminary multiplicative update rule. We then use the orthogonality constraint to solve the multipliers. Substituting the multipliers in the preliminary update rule, we obtain an optimization algorithm which iterates the following multiplicative update rule:

$$W_{ik}^{\text{new}} = W_{ik}\left[\frac{\left(AW + 2\lambda WW^T VW\right)_{ik}}{\left(2\lambda VW + WW^T AW\right)_{ik}}\right]^{1/4} \tag{5}$$

where $V$ is a diagonal matrix with $V_{ii} = \sum_l W_{il}^2$.

**Theorem 1.** $\mathcal{L}(W^{new}, \Lambda) \leq \mathcal{L}(W, \Lambda)$ *for* $\Lambda = \frac{1}{2}W^T\left(\frac{\partial \mathcal{J}}{\partial W}\right)$.

The proof is given the appendix. Note that $\mathcal{J}(W)$ does not necessarily decrease after each iteration. Instead, the monotonicity stated in the theorem justifies that the above algorithm jointly minimizes the $\mathcal{J}(W)$ and drives $W$ towards the manifold defined by the orthogonality constraint. After $W$ converges, we discretize it and obtain the cluster indicator matrix $U$.

It is a crucial observation that the update rule Eq. (5) requires only the product of $(I - \alpha Q)^{-1}$ with a low-rank matrix instead of $A$ itself. We can thus avoid expensive computation and storage of large smoothed similarity matrix. There is an iterative and more scalable way to calculate $F = (I - \alpha Q)^{-1}W$ [29]. See the *IterativeSolver* function in Algorithm 1. In practice, the calculation for $F$ usually converges nicely within 100 iterations. The same technique can be applied to calculating the normalizing factor $c$ in Eq. (2), using $e = [1, 1, \ldots, 1]$ instead of $W$. The resulting algorithm for optimization w.r.t. $W$ is summarized in Algorithm 1. Matlab codes can be found in [1].

## 3.3 Initialization

Most state-of-the-art clustering methods involve non-convex optimization objectives and thus only return local optima in general. This is also the case for our algorithm. To achieve a better local

optimum, a clustering algorithm should start from one or more relatively considerate initial guesses. Different strategies for choosing the starting point can be classified into the following levels, sorted by their computational cost:

**Level-0: (random-init)** The starting relaxed indicator matrix is filled by randomly generated numbers.

**Level-1: (simple-init)** The starting matrix is the result of a cheap clustering method, e.g. Normalized Cut or k-means, plus a small perturbation.

**Level-2: (family-init)** The initial guesses are results of the methods in a parameterized family. Typical examples include various regularization extents or Bayesian priors with different hyperparameters (see e.g. [25]).

**Level-3: (meta-init)** The initial guesses can come from methods of various principles. Each initialization method runs only once.

**Level-4: (meta-co-init)** Same as Level-3 except that clustering methods provide initialization for each other. A method can serve initialization multiple times if it finds a better local minimum. The whole procedure stops when each of the involved methods fails to find better local optimum (see e.g. [10]).

Some methods are not sensitive to initializations but tend to return less accurate clustering. On the other hand, some other methods can find more accurate results but require comprehensive initialization. A preferable clustering method should achieve high accuracy with cheap initialization. As we shall see, the proposed NMFR algorithm can attain satisfactory clustering accuracy with only simple initialization (Level-1).

## 4   Experiments

We have compared our method against a variety of state-of-the-art clustering methods, including Projective NMF [23], Nonnegative Spectral Cut (NSC) [8], (symmetric) Orthogonal NMF (ONMF) [11], Left-Stochastic matrix Decomposition (LSD) [2], Data-Cluster-Data decomposition (DCD) [25], as well as classical Normalized Cut (Ncut) [21]. We also selected two recent clustering methods beyond NMF: 1-Spectral (1Spec) [14] which uses balanced graph cut, and Interaction Component Model (ICM) [22] which is the symmetric version of topic model [3].

We used default settings in the compared methods. For 1Spec, we used ratio Cheeger cut. For ICM, the hyper-parameters for Dirichlet processes prior are updated by Minka's learning method [19]. The other NMF-type methods that use multiplicative updates were run with 10,000 iterations to guarantee convergence. For our method, we trained $W$ by using Algorithm 1 for each candidate $\alpha \in \{0.1, 0.2, 0.3, 0.4, 0.5, 0.6, 0.7, 0.8, 0.9, 0.99\}$ when $n \leq 8000$. The best $\alpha$ and the corresponding clustering result were then obtained by minimizing $\|A - bWW^T\|_F^2$ with a suitable positive scalar $b$. Here we set $b = 2\lambda$ using the heuristic that the penalty term in gradient can be interpreted as removal of diagonal effect of approximating matrix. When $\lambda = \frac{1}{2r}$, we obtain $b = 1/r$. The new clustering method is not very sensitive to the choice of $\alpha$ for large-scale datasets. We simply used $\alpha = 0.8$ in experiments when $n > 8000$. All methods except Ncut, 1Spec, and ICM were initialized by Normalized Cut. That is, their starting point was the Ncut cluster indicator matrix plus a small constant 0.2 to all entries.

We have compared the above methods on clustering various datasets. The domains of the datasets range from network, text, biology, image, etc. All datasets are publicly available on the Internet. The data sources and statistics are given in the supplemental document. We constructed symmetrized $K$-NN graphs from the multivariate data, where $K = 5$ for the 30 smallest datasets, text datasets, *PROTEIN* and *SEISMIC* datasets, while $K = 10$ for the remaining datasets. Following [25], we extract the scattering features [18] for images before calculating the $K$-NN graph. We used Tf-Idf features for text data. The adjacency matrices of network data were symmetrized. The clustering performance is evaluated by cluster $purity = \frac{1}{n}\sum_{k=1}^{r}\max_{1 \leq l \leq r} n_k^l$, where $n_k^l$ is the number of data samples in the cluster $k$ that belong to ground-truth class $l$. A larger purity in general corresponds to a better clustering result.

The resulting purities are shown in Table 1, where the rows are ordered by dataset size. We can see that our method has much better performance than the other methods. NMFR wins 36 out of 46

Table 1: Clustering purities for the compared methods on various datasets. Boldface numbers indicate the best in each row.

| Dataset | Size | Ncut | PNMF | NSC | ONMF | PLSI | LSD | 1Spec | ICM | DCD | NMFR |
|---|---|---|---|---|---|---|---|---|---|---|---|
| STRIKE | 24 | 0.96 | **1.00** | 0.96 | **1.00** | 0.96 | **1.00** | **1.00** | 0.58 | 0.96 | 0.96 |
| KOREA | 35 | **1.00** | 0.94 | 0.71 | **1.00** | **1.00** | **1.00** | 0.71 | 0.66 | 0.97 | **1.00** |
| AMLALL | 38 | **0.92** | **0.92** | **0.92** | **0.92** | **0.92** | **0.92** | **0.92** | 0.50 | **0.92** | 0.89 |
| DUKE | 44 | 0.52 | 0.52 | 0.52 | 0.52 | **0.70** | **0.70** | 0.52 | 0.52 | 0.52 | **0.70** |
| HIGHSCHOOL | 60 | 0.83 | 0.82 | 0.83 | 0.82 | 0.83 | 0.82 | 0.82 | 0.82 | 0.83 | **0.95** |
| KHAN | 83 | 0.57 | **0.60** | 0.55 | **0.60** | 0.55 | 0.52 | 0.58 | 0.49 | 0.55 | 0.51 |
| POLBOOKS | 105 | 0.78 | 0.78 | 0.81 | 0.77 | 0.78 | 0.78 | **0.83** | 0.78 | 0.79 | 0.79 |
| FOOTBALL | 115 | **0.93** | **0.93** | **0.93** | **0.93** | **0.93** | **0.93** | 0.90 | **0.93** | **0.93** | **0.93** |
| IRIS | 150 | 0.90 | **0.93** | 0.90 | 0.92 | 0.91 | 0.75 | 0.91 | 0.53 | 0.91 | 0.91 |
| CANCER | 198 | 0.53 | **0.54** | 0.53 | 0.53 | **0.54** | 0.53 | 0.51 | 0.53 | **0.54** | 0.52 |
| SPECT | 267 | **0.79** | **0.79** | **0.79** | **0.79** | **0.79** | **0.79** | **0.79** | **0.79** | **0.79** | **0.79** |
| ROSETTA | 300 | **0.77** | **0.77** | **0.77** | **0.77** | **0.77** | **0.77** | **0.77** | **0.77** | **0.77** | **0.77** |
| ECOLI | 327 | 0.79 | 0.78 | 0.79 | 0.78 | 0.80 | 0.68 | **0.83** | 0.78 | 0.80 | 0.79 |
| IONOSPHERE | 351 | 0.69 | 0.69 | **0.70** | 0.69 | 0.69 | 0.64 | 0.69 | 0.69 | 0.69 | 0.68 |
| ORL | 400 | 0.81 | 0.82 | 0.82 | 0.82 | **0.83** | 0.81 | 0.80 | 0.19 | **0.83** | **0.83** |
| UMIST | 575 | 0.68 | 0.64 | 0.68 | 0.66 | 0.69 | 0.68 | **0.74** | 0.15 | 0.69 | 0.72 |
| WDBC | 683 | **0.65** | **0.65** | **0.65** | **0.65** | **0.65** | **0.65** | **0.65** | **0.65** | **0.65** | **0.65** |
| DIABETES | 768 | **0.65** | **0.65** | **0.65** | **0.65** | **0.65** | **0.65** | **0.65** | **0.65** | **0.65** | **0.65** |
| VOWEL | 1.0K | 0.36 | 0.35 | 0.36 | 0.30 | 0.36 | 0.34 | 0.20 | 0.15 | 0.36 | **0.37** |
| MED | 1.0K | 0.53 | 0.54 | 0.54 | 0.54 | 0.54 | 0.55 | 0.50 | 0.33 | 0.55 | **0.56** |
| PIE | 1.2K | 0.67 | 0.66 | 0.68 | 0.66 | 0.68 | 0.69 | 0.64 | 0.12 | 0.68 | **0.74** |
| YALEB | 1.3K | 0.45 | 0.42 | 0.46 | 0.41 | **0.51** | 0.50 | 0.37 | 0.10 | **0.51** | **0.51** |
| TERROR | 1.3K | 0.45 | 0.45 | 0.46 | 0.46 | 0.46 | 0.45 | 0.44 | 0.34 | 0.45 | **0.49** |
| ALPHADIGS | 1.4K | 0.49 | 0.45 | 0.49 | 0.44 | 0.49 | 0.49 | 0.48 | 0.10 | 0.50 | **0.51** |
| COIL-20 | 1.4K | 0.79 | 0.71 | 0.79 | 0.65 | 0.79 | 0.75 | 0.77 | 0.11 | 0.79 | **0.81** |
| YEAST | 1.5K | 0.53 | 0.53 | 0.54 | 0.52 | 0.53 | 0.52 | 0.54 | 0.34 | 0.52 | **0.55** |
| SEMEION | 1.6K | 0.83 | 0.87 | 0.83 | 0.85 | 0.85 | 0.89 | 0.82 | 0.13 | 0.85 | **0.94** |
| FAULTS | 1.9K | 0.40 | 0.39 | 0.40 | 0.39 | 0.40 | 0.40 | 0.38 | 0.38 | **0.41** | 0.39 |
| SEG | 2.3K | 0.61 | 0.51 | 0.61 | 0.53 | 0.61 | 0.64 | 0.55 | 0.32 | 0.61 | **0.73** |
| ADS | 2.4K | **0.84** | **0.84** | **0.84** | **0.84** | **0.84** | **0.84** | **0.84** | **0.84** | **0.84** | **0.84** |
| CORA | 2.7K | 0.38 | 0.37 | 0.37 | 0.37 | 0.44 | 0.46 | 0.36 | 0.30 | 0.44 | **0.47** |
| MIREX | 3.1K | 0.41 | 0.40 | 0.42 | 0.38 | 0.41 | 0.38 | 0.12 | 0.27 | 0.18 | **0.43** |
| CITESEER | 3.3K | 0.24 | 0.31 | 0.23 | 0.31 | 0.36 | 0.36 | 0.22 | 0.41 | 0.35 | **0.44** |
| WEBKB4 | 4.2K | 0.40 | 0.39 | 0.40 | 0.39 | 0.49 | 0.51 | 0.39 | 0.48 | 0.51 | **0.63** |
| 7SECTORS | 4.6K | 0.25 | 0.27 | 0.25 | 0.25 | 0.29 | 0.26 | 0.25 | 0.28 | 0.28 | **0.34** |
| SPAM | 4.6K | 0.61 | 0.61 | 0.61 | 0.61 | 0.65 | 0.68 | 0.61 | 0.61 | 0.67 | **0.69** |
| CURETGREY | 5.6K | 0.26 | 0.22 | 0.26 | 0.21 | 0.26 | 0.21 | 0.22 | 0.11 | 0.27 | **0.28** |
| OPTDIGITS | 5.6K | 0.92 | 0.90 | 0.92 | 0.90 | 0.93 | 0.92 | 0.87 | 0.90 | 0.92 | **0.98** |
| GISETTE | 7.0K | 0.90 | 0.52 | 0.93 | 0.51 | 0.93 | 0.93 | 0.93 | 0.62 | 0.93 | **0.94** |
| REUTERS | 8.3K | **0.77** | 0.74 | 0.76 | 0.72 | 0.76 | 0.75 | 0.63 | 0.71 | 0.76 | **0.77** |
| RCV1 | 9.6K | 0.33 | 0.35 | 0.32 | 0.31 | 0.37 | 0.48 | 0.31 | 0.38 | 0.36 | **0.54** |
| PENDIGITS | 11K | 0.80 | 0.82 | 0.80 | 0.77 | 0.80 | 0.86 | 0.82 | 0.52 | 0.80 | **0.87** |
| PROTEIN | 18K | 0.46 | 0.46 | 0.46 | 0.46 | 0.46 | 0.46 | 0.46 | 0.46 | 0.46 | **0.50** |
| 20NEWS | 20K | 0.25 | 0.33 | 0.21 | 0.31 | 0.31 | 0.32 | 0.07 | 0.23 | 0.31 | **0.63** |
| MNIST | 70K | 0.77 | 0.87 | 0.79 | 0.73 | 0.79 | 0.76 | 0.88 | 0.95 | 0.82 | **0.97** |
| SEISMIC | 99K | 0.52 | 0.50 | 0.51 | 0.50 | 0.52 | 0.54 | 0.51 | 0.50 | 0.52 | **0.59** |

selected clustering tasks. Our method is especially superior for large-scale data in a curved manifold, for example, *OPTDIGITS* and *MNIST*. Note that cluster purity can be regarded as classification accuracy if we have a few labeled data samples to remove ambiguity between clusters and classes. In this sense, the resulting purities for such manifold data are even comparable to the state-of-the-art supervised classification results. Compared with the DCD results which require Level-2 family initialization (see [25]), NMFR only needs Level-1 simple initialization. In addition, NMFR also brings remarkable improvement for datasets beyond digit or letter recognition, for example, the text data *RCV1*, *20NEWS*, protein data *PROTEIN* and sensor data *SEISMIC*. Furthermore, it is worth to notice that our method has more robust performance over various datasets compared with other approaches. Even for some small datasets where NMFR is not the winner, its cluster purities are still close to the best.

# 5   Conclusions

We have presented a new NMF method using random walk for clustering. Our work includes two major contributions: (1) we have shown that NMF approximation using least square error should be applied on smoothed similarities; the smoothing accompanied with a novel regularization can often significantly outperform spectral clustering; (2) the smoothing is realized in an implicit and scalable way. Extensive empirical study has shown that our method can often improve clustering accuracy remarkably given simple initialization.

Some issues could be included in the future work. Here we only discuss a certain type of smoothing by random walk, while the proposed method could be extended by using other types of smoothing, e.g. diffusion kernels, where scalable optimization could also be developed by using a similar iterative subroutine. Moreover, the smoothing brings improved clustering accuracy but at the cost of increased running time. Algorithms that are more efficient in both time and space should be further investigated. In addition, the approximated matrix could also be learnable. In current experiments, we used constant $K$-NN graphs as input for fair comparison, which could be replaced by a more comprehensive graph construction method (e.g. [28, 12, 17]).

# 6   Acknowledgement

This work was financially supported by the Academy of Finland (Finnish Center of Excellence in Computational Inference Research COIN, grant no 251170; Zhirong Yang additionally by decision number 140398).

# Appendix: proof of Theorem 1

The proof follows the Majorization-Minimization development procedure in [26]. We use $W$ and $\widetilde{W}$ to distinguish the current estimate and the variable, respectively.

Given a real-valued matrix $B$, we can always decompose it into two nonnegative parts such that $B = B^+ - B^-$, where $B_{ij}^+ = (|B_{ij}| + B_{ij})/2$ and $B_{ij}^- = (|B_{ij}| - B_{ij})/2$. In this way we decompose $\Lambda = \Lambda^+ - \Lambda^-$ and $\left.\frac{\partial \mathcal{J}(\widetilde{W})}{\partial \widetilde{W}}\right|_{\widetilde{W}=W} \equiv \nabla = \nabla^+ - \nabla^-$, where $\nabla^+ = 4\lambda VW$ and $\nabla^- = 2AW$.

(**Majorization**) Up to some additive constant,

$$\widetilde{\mathcal{J}}(\widetilde{W}, \Lambda)$$

$$\leq -2\mathrm{Tr}\left(\widetilde{W}^T A W\right) + \lambda \sum_{ik}\left(\sum_l W_{il}^2\right)\frac{\widetilde{W}^4}{W_{ik}^2} + \sum_{ik}\frac{\widetilde{W}_{ik}^2}{W_{ik}}\left(\Lambda^+ W\right)_{ik} - 2\mathrm{Tr}\left(\widetilde{W}^T \Lambda^- W\right)$$

$$\leq -2\mathrm{Tr}\left(\widetilde{W}^T A W\right) + \lambda \sum_{ik}\left(\sum_l W_{il}^2\right)\frac{\widetilde{W}^4}{W_{ik}^2} + \sum_{ik}\frac{W_{ik}\left(\Lambda^+ W\right)_{ik}}{2}\left(\frac{\widetilde{W}_{ik}}{W_{ik}}\right)^4 - 2\mathrm{Tr}\left(\widetilde{W}^T \Lambda^- W\right)$$

$$\equiv G(\widetilde{W}, W),$$

where the first inequality is by standard convex-concave procedure, and the second upper bound is due to the inequality $\frac{z^a - 1}{a} \leq \frac{z^b - 1}{b}$ for $z > 0$ and $a < b$.

(**Minimization**) Setting $\partial G(\widetilde{W}, \Lambda)/\partial \widetilde{W}_{ik} = 0$ gives

$$W_{ik}^{\mathrm{new}} = W_{ik}\left[\frac{\left(\nabla^- + 2\mathbf{W}\mathbf{\Lambda}^+\right)_{ik}}{\left(\nabla^+ + 2\mathbf{W}\mathbf{\Lambda}^-\right)_{ik}}\right]^{1/4}. \tag{6}$$

Zeroing $\partial \mathcal{L}(W, \Lambda)/\partial W = 0$ gives $2W\Lambda = \nabla^+ - \nabla^-$. Using $W^T W = I$, we obtain $\Lambda = \frac{1}{2}W^T(\nabla^+ - \nabla^-)$, i.e. $2W\Lambda^+ = WW^T\nabla^+$ and $2W\Lambda^- = WW^T\nabla^-$. Inserting these into Eq. (6), we obtain update rule in Eq. (5).

# References

[1] http://users.ics.aalto.fi/rozyang/nmfr/index.shtml.

[2] R. Arora, M. Gupta, A. Kapila, and M. Fazel. Clustering by left-stochastic matrix factorization. In *ICML*, 2011.

[3] D. Blei, A. Y. Ng, and M. I. Jordan. Latent dirichlet allocation. *Journal of Machine Learning Research*, 3:993–1022, 2001.

[4] Deng Cai, Xiaofei He, Jiawei Han, and Thomas S. Huang. Graph regularized non-negative matrix factorization for data representation. *IEEE Transactions on Pattern Analysis and Machine Intelligence*, 33(8):1548–1560, 2011.

[5] A. Cichocki, S. Cruces, and S. Amari. Generalized alpha-beta divergences and their application to robust nonnegative matrix factorization. *Entropy*, 13:134–170, 2011.

[6] I. Dhillon, Y. Guan, and B. Kulis. Kernel k-means, spectral clustering and normalized cuts. In *KDD*, 2004.

[7] C. Ding, X. He, and H. D. Simon. On the equivalence of nonnegative matrix factorization and spectral clustering. In *ICDM*, 2005.

[8] C. Ding, T. Li, and M. I. Jordan. Nonnegative matrix factorization for combinatorial optimization: Spectral clustering, graph matching, and clique finding. In *ICDM*, 2008.

[9] C. Ding, T. Li, and M. I. Jordan. Convex and semi-nonnegative matrix factorizations. *IEEE Transactions on Pattern Analysis and Machine Intelligence*, 32(1):45–55, 2010.

[10] C. Ding, T. Li, and W. Peng. On the equivalence between non-negative matrix factorization and probabilistic laten semantic indexing. *Computational Statistics and Data Analysis*, 52(8):3913–3927, 2008.

[11] C. Ding, T. Li, W. Peng, and H. Park. Orthogonal nonnegative matrix t-factorizations for clustering. In *SIGKDD*, 2006.

[12] E. Elhamifar and R. Vidal. Sparse manifold clustering and embedding. In *NIPS*, 2011.

[13] Z. He, S. Xie, R. Zdunek, G. Zhou, and A. Cichocki. Symmetric nonnegative matrix factorization: Algorithms and applications to probabilistic clustering. *IEEE Transactions on Neural Networks*, 22(12):2117–2131, 2011.

[14] M. Hein and T. Bühler. An inverse power method for nonlinear eigenproblems with applications in 1-Spectral clustering and sparse PCA. In *NIPS*, 2010.

[15] D. D. Lee and H. S. Seung. Algorithms for non-negative matrix factorization. In *NIPS*, 2000.

[16] C.-J. Lin. Projected gradient methods for non-negative matrix factorization. *Neural Computation*, 19:2756–2779, 2007.

[17] M. Maier, U. von Luxburg, and M. Hein. How the result of graph clustering methods depends on the construction of the graph. *ESAIM: Probability & Statistics*, 2012. in press.

[18] S. Mallat. Group invariant scattering. ArXiv e-prints, 2011.

[19] T. Minka. Estimating a dirichlet distribution, 2000.

[20] A. Ng, M. Jordan, and Y. Weiss. On spectral clustering: Analysis and an algorithm. In *NIPS*, 2001.

[21] J. Shi and J. Malik. Normalized cuts and image segmentation. *IEEE Transactions on Pattern Analysis and Machine Intelligence*, 22(8):888–905, 2000.

[22] J. Sinkkonen, J. Aukia, and S. Kaski. Component models for large networks. ArXiv e-prints, 2008.

[23] Z. Yang and E. Oja. Linear and nonlinear projective nonnegative matrix factorization. *IEEE Transaction on Neural Networks*, 21(5):734–749, 2010.

[24] Z. Yang and E. Oja. Unified development of multiplicative algorithms for linear and quadratic nonnegative matrix factorization. *IEEE Transactions on Neural Networks*, 22(12):1878–1891, 2011.

[25] Z. Yang and E. Oja. Clustering by low-rank doubly stochastic matrix decomposition. In *ICML*, 2012.

[26] Z. Yang and E. Oja. Quadratic nonnegative matrix factorization. *Pattern Recognition*, 45(4):1500–1510, 2012.

[27] R. Zass and A. Shashua. A unifying approach to hard and probabilistic clustering. In *ICCV*, 2005.

[28] L. Zelnik-Manor and P. Perona. Self-tuning spectral clustering. In *NIPS*, 2004.

[29] D. Zhou, O. Bousquet, T. Lal, J. Weston, and B. Schölkopf. Learning with local and global consistency. In *NIPS*, 2003.

